# An Oscillatory Correlation Framework for Computational Auditory Scene Analysis

**Guy J. Brown**
Department of Computer Science
University of Sheffield
Regent Court, 211 Portobello Street,
Sheffield S1 4DP, UK
*Email: g.brown@dcs.shef.ac.uk*

**DeLiang L. Wang**
Department of Computer and Information
Science and Centre for Cognitive Science
The Ohio State University
Columbus, OH 43210-1277, USA
*Email: dwang@cis.ohio-state.edu*

## Abstract

A neural model is described which uses oscillatory correlation to segregate speech from interfering sound sources. The core of the model is a two-layer neural oscillator network. A sound stream is represented by a synchronized population of oscillators, and different streams are represented by desynchronized oscillator populations. The model has been evaluated using a corpus of speech mixed with interfering sounds, and produces an improvement in signal-to-noise ratio for every mixture.

## 1 Introduction

Speech is seldom heard in isolation: usually, it is mixed with other environmental sounds. Hence, the auditory system must parse the acoustic mixture reaching the ears in order to retrieve a description of each sound source, a process termed *auditory scene analysis* (ASA) [2]. Conceptually, ASA may be regarded as a two-stage process. The first stage (which we term 'segmentation') decomposes the acoustic stimulus into a collection of sensory elements. In the second stage ('grouping'), elements that are likely to have arisen from the same environmental event are combined into a perceptual structure called a *stream*. Streams may be further interpreted by higher-level cognitive processes.

Recently, there has been a growing interest in the development of computational systems that mimic ASA [4], [1], [5]. Such computational auditory scene analysis (CASA) systems are inspired by auditory function but do not model it closely; rather, they employ symbolic search or high-level inference engines. Although the performance of these systems is encouraging, they are no match for the abilities of a human listener; also, they tend to be complex and computationally intensive. In short, CASA currently remains an unsolved problem for real-time applications such as automatic speech recognition.

Given that human listeners can segregate concurrent sounds with apparent ease, computational systems that are more closely modelled on the neurobiological mechanisms of hearing may offer a performance advantage over existing CASA systems. This observation – together with a desire to understand the neurobiological basis of ASA – has led some investigators to propose neural network models of ASA. Most recently, Brown and Wang [3] have given an account of concurrent vowel separation based on *oscillatory correlation*. In this framework, oscillators that represent a perceptual stream are synchronized (phase locked with zero phase lag), and are desynchronized from oscillators that represent different streams [8]. Evidence for the oscillatory correlation theory comes from neurobiological studies which report synchronised oscillations in the auditory, visual and olfactory cortices (see [10] for a review).

In this paper, we propose a neural network model that uses oscillatory correlation as the underlying neural mechanism for ASA; streams are formed by synchronizing oscillators in a two-dimensional time-frequency network. The model is evaluated on a task that involves the separation of two time-varying sounds. It therefore extends our previous study [3], which only considered the segregation of vowel sounds with static spectra.

## 2   Model description

The input to the model consists of a mixture of speech and an interfering sound source, sampled at a rate of 16 kHz with 16 bit resolution. This input signal is processed in four stages described below (see [10] for a detailed account).

### 2.1  Peripheral auditory processing

Peripheral auditory frequency selectivity is modelled using a bank of 128 gammatone filters with center frequencies equally distributed on the equivalent rectangular bandwidth (ERB) scale between 80 Hz and 5 kHz [1]. Subsequently, the output of each filter is processed by a model of inner hair cell function. The output of the hair cell model is a probabilistic representation of auditory nerve firing activity.

### 2.2  Mid-level auditory representations

Mechanisms similar to those underlying pitch perception can contribute to the perceptual separation of sounds that have different fundamental frequencies (F0s) [3]. Accordingly, the second stage of the model extracts periodicity information from the simulated auditory nerve firing patterns. This is achieved by computing a running autocorrelation of the auditory nerve activity in each channel, forming a representation known as a *correlogram* [1], [5]. At time step $j$, the autocorrelation $A(i,j,\tau)$ for channel $i$ with time lag $\tau$ is given by:

$$A(i, j, \tau) = \sum_{k=0}^{K-1} r(i, j-k)r(i, j-k-\tau)w(k) \tag{1}$$

Here, $r$ is the output of the hair cell model and $w$ is a rectangular window of width $K$ time steps. We use $K = 320$, corresponding to a window width of 20 ms. The autocorrelation lag $\tau$ is computed in $L$ steps of the sampling period between 0 and $L$-1; we use $L = 201$, corresponding to a maximum delay of 12.5 ms. Equation (1) is computed for $M$ time frames, taken at 10 ms intervals (i.e., at intervals of 160 steps of the time index $j$).

For periodic sounds, a characteristic 'spine' appears in the correlogram which is centered on the lag corresponding to the stimulus period (Figure 1A). This pitch-related structure can be emphasized by forming a 'pooled' correlogram $s(j,\tau)$, which exhibits a prominent peak at the delay corresponding to perceived pitch:

$$s(j, \tau) = \sum_{i=1}^{N} A(i, j, \tau) \tag{2}$$

It is also possible to extract harmonics and formants from the correlogram, since frequency channels that are excited by the same acoustic component share a similar pattern of periodicity. Bands of coherent periodicity can be identified by cross-correlating adjacent correlogram channels; regions of high correlation indicate a harmonic or formant [1]. The cross-correlation $C(i,j)$ between channels $i$ and $i+1$ at time frame $j$ is defined as:

$$C(i, j) = \frac{1}{L} \sum_{\tau=0}^{L-1} \hat{A}(i, j, \tau)\hat{A}(i+1, j, \tau) \quad (1 \le i \le N-1) \tag{3}$$

Here, $\hat{A}(i, j, \tau)$ is the autocorrelation function of (1) which has been normalized to have zero mean and unity variance. A typical cross-correlation function is shown in Figure 1A.

## 2.3 Neural oscillator network: overview

Segmentation and grouping take place within a two-layer oscillator network (Figure 1B). The basic unit of the network is a single oscillator, which is defined as a reciprocally connected excitatory variable $x$ and inhibitory variable $y$ [7]. Since each layer of the network takes the form of a time-frequency grid, we index each oscillator according to its frequency channel ($i$) and time frame ($j$):

$$\dot{x}_{ij} = 3x_{ij} - x_{ij}^3 + 2 - y_{ij} + I_{ij} + S_{ij} + \rho \qquad (4a)$$

$$\dot{y}_{ij} = \varepsilon(\gamma(1 + \tanh(x_{ij}/\beta)) - y_{ij}) \qquad (4b)$$

Here, $I_{ij}$ represents external input to the oscillator, $S_{ij}$ denotes the coupling from other oscillators in the network, $\varepsilon$, $\gamma$ and $\beta$ are parameters, and $\rho$ is the amplitude of a Gaussian noise term. If coupling and noise are ignored and $I_{ij}$ is held constant, (4) defines a relaxation oscillator with two time scales. The $x$-nullcline, i.e. $\dot{x}_{ij} = 0$, is a cubic function and the $y$-nullcline is a sigmoid function. If $I_{ij} > 0$, the two nullclines intersect only at a point along the middle branch of the cubic with $\beta$ chosen small. In this case, the oscillator exhibits a stable limit cycle for small values of $\varepsilon$, and is referred to as *enabled*. The limit cycle alternates between *silent* and *active* phases of near steady-state behaviour. Compared to motion within each phase, the alternation between phases takes place rapidly, and is referred to as *jumping*. If $I_{ij} < 0$, the two nullclines intersect at a stable fixed point. In this case, no oscillation occurs. Hence, oscillations in (4) are stimulus-dependent.

## 2.4 Neural oscillator network: segment layer

In the first layer of the network, *segments* are formed – blocks of synchronised oscillators that trace the evolution of an acoustic component through time and frequency. The first layer is a two-dimensional time-frequency grid of oscillators with a global inhibitor (see Figure 1B). The coupling term $S_{ij}$ in (4a) is defined as

$$S_{ij} = \sum_{kl \in N(i, j)} W_{ij,kl} H(x_{kl} - \theta_x) - W_z H(z - \theta_z) \qquad (5)$$

where $H$ is the Heaviside function (i.e., $H(x) = 1$ for $x \geq 0$, and zero otherwise), $W_{ij,kl}$ is the connection weight from an oscillator $(i,j)$ to an oscillator $(k,l)$ and $N(i,j)$ is the four nearest neighbors of $(i,j)$. The threshold $\theta_x$ is chosen so that an oscillator has no influence on its

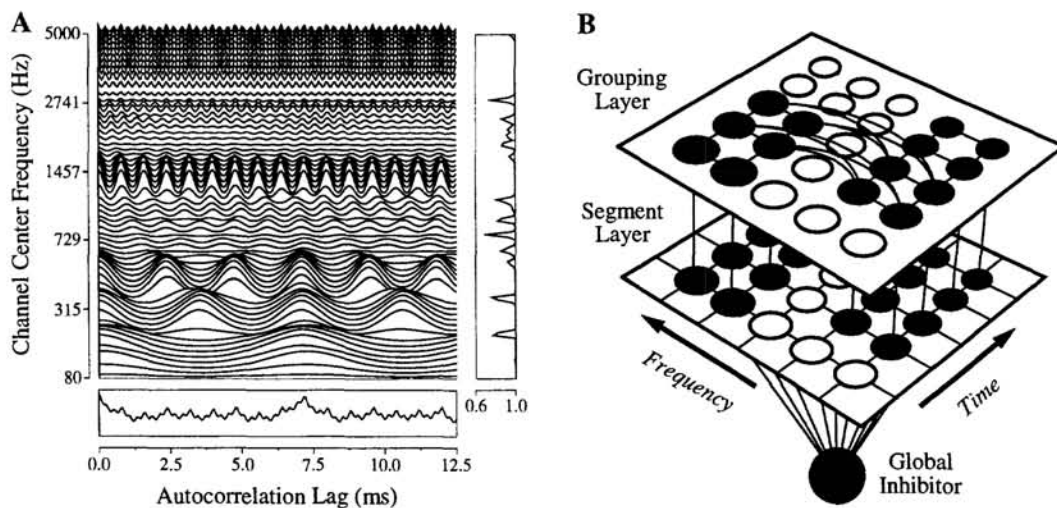

Figure 1: A. Correlogram of a mixture of speech and trill telephone, taken 450 ms after the start of the stimulus. The pooled correlogram is shown in the bottom panel, and the cross-correlation function is shown on the right. B. Structure of the two-layer oscillator network.

neighbors unless it is in the active phase. The weight of neighboring connections along the time axis is uniformly set to 1. The connection weight between an oscillator $(i,j)$ and its vertical neighbor $(i+1,j)$ is set to 1 if $C(i,j)$ exceeds a threshold $\theta_c$; otherwise it is set to 0. $W_z$ is the weight of inhibition from the global inhibitor $z$, defined as

$$\dot{z} = \sigma_\infty - z \tag{6}$$

where $\sigma_\infty = 1$ if $x_{ij} \geq \theta_z$ for at least one oscillator $(i,j)$, and $\sigma_\infty = 0$ otherwise. Hence $\theta_z$ is a threshold. If $\sigma_\infty = 1$, $z \rightarrow 1$.

Small segments may form which do not correspond to perceptually significant acoustic components. In order to remove these noisy fragments, we introduce a lateral potential $p_{ij}$ for oscillator $(i,j)$, defined as [11]:

$$\dot{p}_{ij} = (1 - p_{ij})H\left[\sum_{kl \in N_p(i,j)} H(x_{kl} - \theta_x) - \theta_p\right] - \varepsilon p_{ij} \tag{7}$$

Here, $\theta_p$ is a threshold. $N_p(i,j)$ is called the potential neighborhood of $(i,j)$, which is chosen to be $(i,j-1)$ and $(i,j+1)$. If both neighbors of $(i,j)$ are active, $p_{ij}$ approaches 1 on a fast time scale; otherwise, $p_{ij}$ relaxes to 0 on a slow time scale determined by $\varepsilon$.

The lateral potential plays its role by gating the input to an oscillator. More specifically, we replace (4a) with

$$\dot{x}_{ij} = 3x_{ij} - x_{ij}^3 + 2 - y_{ij} + I_{ij}H(p_{ij} - \theta) + S_{ij} + \rho \tag{4a'}$$

With $p_{ij}$ initialized to 1, it follows that $p_{ij}$ will drop below the threshold $\theta$ unless the oscillator $(i,j)$ receives excitation from its entire potential neighborhood. Given our choice of neighborhood in (5), this implies that a segment must extend for at least three consecutive time frames. Oscillators that are stimulated but cannot maintain a high potential are relegated to a discontiguous 'background' of noisy activity.

An oscillator $(i,j)$ is stimulated if its corresponding input $I_{ij} > 0$. Oscillators are stimulated only if the energy in their corresponding correlogram channel exceeds a threshold $\theta_a$. It is evident from (1) that the energy in a correlogram channel $i$ at time $j$ corresponds to $A(i,j,0)$; thus we set $I_{ij} = 0.2$ if $A(i,j,0) > \theta_a$, and $I_{ij} = -5$ otherwise.

Figure 2A shows the segmentation of a mixture of speech and trill telephone. The network was simulated by the LEGION algorithm [8], producing 94 segments (each represented by a distinct gray level) plus the background (shown in black). For convenience we show all segments together in Figure 2A, but each actually arises during a unique time interval.

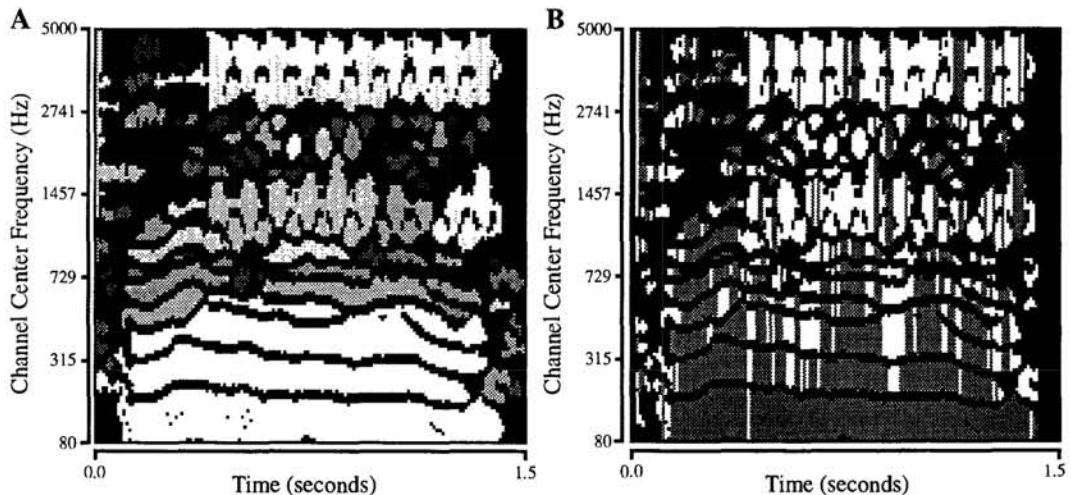

Figure 2: A. Segments formed by the first layer of the network for a mixture of speech and trill telephone. B. Categorization of segments according to F0. Gray pixels represent the set P, and white pixels represent regions that do not agree with the F0.

## 2.5 Neural oscillator network: grouping layer

The second layer is a two-dimensional network of laterally coupled oscillators without global inhibition. Oscillators in this layer are stimulated if the corresponding oscillator in the first layer is stimulated and does not form part of the background. Initially, all oscillators have the same phase, implying that all segments from the first layer are allocated to the same stream. This initialization is consistent with psychophysical evidence suggesting that perceptual fusion is the default state of auditory organisation [2]. In the second layer, an oscillator has the same form as in (4), except that $x_{ij}$ is changed to:

$$\dot{x}_{ij} = 3x_{ij} - x_{ij}^3 + 2 - y_{ij} + I_{ij}[1 + \mu H(p_{ij} - \theta)] + S_{ij} + \rho \qquad (4a'')$$

Here, $\mu$ is a small positive parameter; this implies that an oscillator with a high lateral potential gets a slightly higher external input. We choose $N_p(i,j)$ and $\theta_p$ so that oscillators which correspond to the longest segment from the first layer are the first to jump to the active phase. The longest segment is identified by using the mechanism described in [9].

The coupling term in (4a'') consists of two types of coupling:

$$S_{ij} = S_{ij}^e + S_{ij}^v \qquad (8)$$

Here, $S_{ij}^e$ represents mutual excitation between oscillators within each segment. We set $S_{ij}^e = 4$ if the active oscillators from the same segment occupy more than half of the length of the segment; otherwise $S_{ij}^e = 0.1$ if there is at least one active oscillator from the same segment.

The coupling term $S_{ij}^v$ denotes vertical connections between oscillators corresponding to different frequency channels and different segments, but within the same time frame. At each time frame, an F0 is estimated from the pooled correlogram (2) and this is used to classify frequency channels into two categories: a set of channels, $P$, that are consistent with the F0, and a set of channels that are not (Figure 2B). Given the delay $\tau_m$ at which the largest peak occurs in the pooled correlogram, for each channel $i$ at time frame $j$, $i \in P$ if

$$A(i, j, \tau_m) / A(i, j, 0) > \theta_d \qquad (9)$$

Since $A(i,j,0)$ is the energy in correlogram channel $i$ at time $j$, (9) amounts to classification on the basis of an energy threshold. We use $\theta_d = 0.95$. The delay $\tau_m$ can be found by using a winner-take-all network, although for simplicity we currently apply a maximum selector.

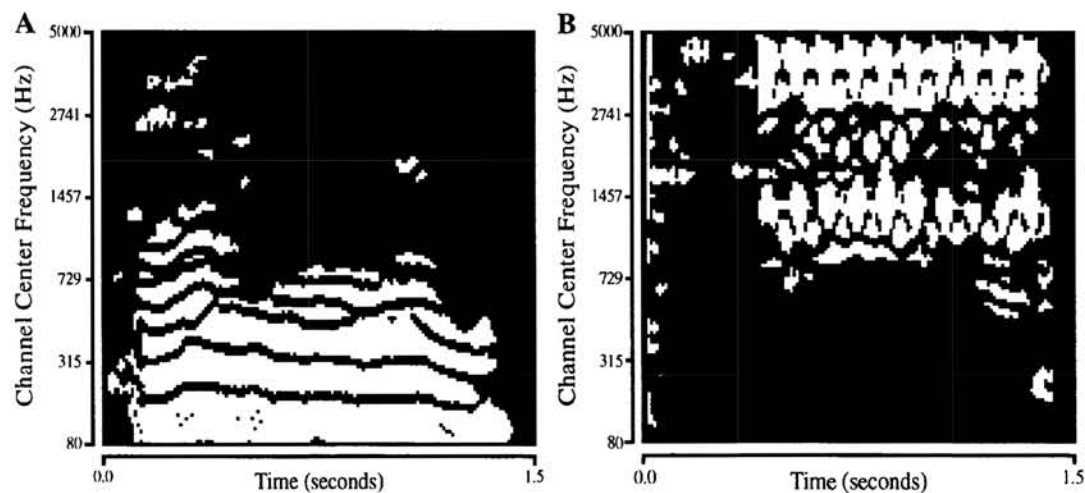

Figure 3: A. Snapshot showing the activity of the second layer shortly after the start of simulation. Active oscillators (white pixels) correspond to the speech stream. B. Another snapshot, taken shortly after A. Active oscillators correspond to the telephone stream.

The F0 classification process operates on channels, rather than segments. As a result, channels within the same segment at a particular time frame may be allocated to different F0 categories. Since segments cannot be decomposed, we enforce a rule that all channels of the same frame within each segment must belong to the same F0 category as that of the majority of channels. After this conformational step, vertical connections are formed such that, at each time frame, two oscillators of different segments have mutual excitatory links if the two corresponding channels belong to the same F0 category; otherwise they have mutual inhibitory links. $S_{ij}^v$ is set to -0.5 if $(i,j)$ receives an input from its inhibitory links; similarly, $S_{ij}^v$ is set to 0.5 if $(i,j)$ receives an input from its vertical excitatory links.

At present, our model has no mechanism for grouping segments that do not overlap in time. Accordingly, we limit operation of the second layer to the time span of the longest segment. After forming lateral connections and trimming by the longest segment, the network is numerically solved using the singular limit method [6].

Figure 3 shows the response of the second layer to the mixture of speech and trill telephone. The figure shows two snapshots of the second layer, where a white pixel indicates an active oscillator and a black pixel indicates a silent oscillator. The network quickly forms two synchronous blocks, which desynchronize from each other. Figure 3A shows a snapshot taken when the oscillator block (stream) corresponding to the segregated speech is in the active phase; Figure 3B shows a subsequent snapshot when the oscillator block corresponding to the trill telephone is in the active phase. Hence, the activity in this layer of the network embodies the result of ASA; the components of an acoustic mixture have been separated using F0 information and represented by oscillatory correlation.

### 2.6 Resynthesis

The last stage of the model is a resynthesis path. Phase-corrected output from the gammatone filterbank is divided into 20 ms sections, overlapping by 10 ms and windowed with a raised cosine. A weighting is then applied to each section, which is unity if the corresponding oscillator is in its active phase, and zero otherwise. The weighted filter outputs are summed across all channels to yield a resynthesized waveform.

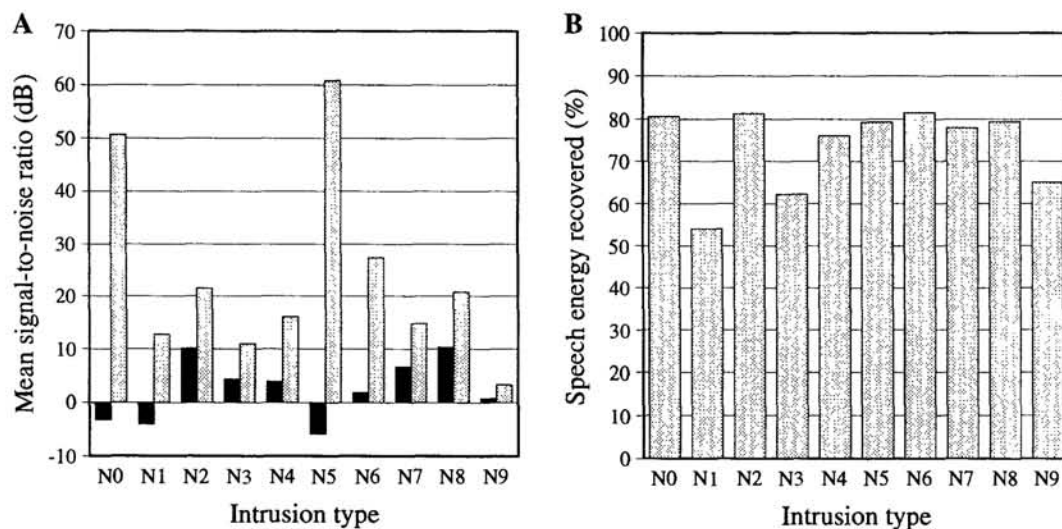

Figure 4: A. SNR before (black bar) and after (grey bar) separation by the model. Results are shown for voiced speech mixed with ten intrusions (N0 = 1 kHz tone; N1 = random noise; N2 = noise bursts; N3 = 'cocktail party' noise; N4 = rock music; N5 = siren; N6 = trill telephone; N7 = female speech; N8 = male speech; N9 = female speech). B. Percentage of speech energy recovered from each mixture after separation by the model.